# Making Templates Rotationally Invariant: An Application to Rotated Digit Recognition

**Shumeet Baluja**
baluja@cs.cmu.edu
Justsystem Pittsburgh Research Center &
School of Computer Science, Carnegie Mellon University

## Abstract

This paper describes a simple and efficient method to make template-based object classification invariant to in-plane rotations. The task is divided into two parts: orientation discrimination and classification. The key idea is to perform the orientation discrimination *before* the classification. This can be accomplished by hypothesizing, in turn, that the input image belongs to each class of interest. The image can then be rotated to maximize its similarity to the training images in each class (these contain the prototype object in an upright orientation). This process yields a set of images, at least one of which will have the object in an upright position. The resulting images can then be classified by models which have been trained with only upright examples. This approach has been successfully applied to two real-world vision-based tasks: rotated handwritten digit recognition and rotated face detection in cluttered scenes.

## 1 Introduction

Rotated text is commonly used in a variety of situations, ranging from advertisements, logos, official post-office stamps, and headlines in magazines, to name a few. For examples, see Figure 1. We would like to be able to recognize these digits or characters, regardless of their rotation.

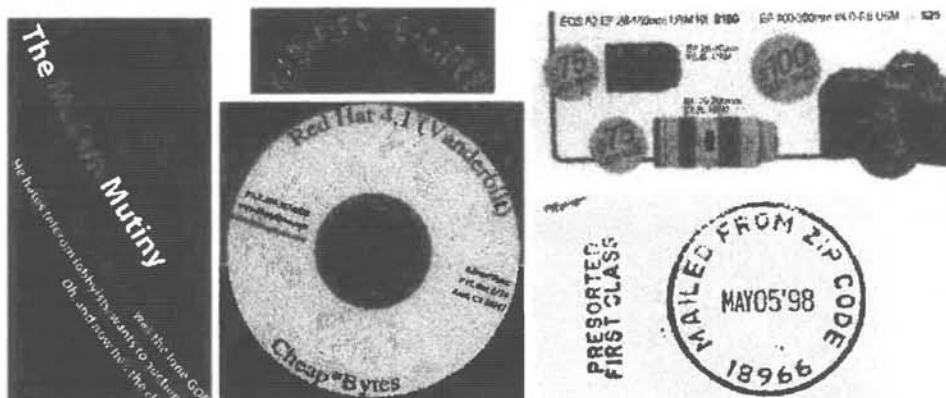

**Figure 1:** Common examples of images which contain text that is not axis aligned include logos, post-office stamps, magazine headlines and consumer advertisements.

The focus of this paper is on the recognition of rotated digits. The simplest method for creating a system which can recognize digits rotated within the image-plane is to employ existing systems which are designed only for upright digit recognition [Le Cun *et al.*, 1990][Le Cun *et al.*, 1995a][Le Cun *et al.*, 1995b][Lee, 1991][Guyon *et al.*, 1989]. By repeatedly rotating the input image by small increments and applying the recognition system at each rotation, the digit will eventually be recognized. As will be discussed in this paper, besides being extremely computationally expensive, this approach is also error-prone. Because the classification of each digit must occur in many orientations, the likelihood of an incorrect match is high.

The procedure presented in this paper to make templates rotationally invariant is significantly faster and more accurate than the one described above. Detailed descriptions of the procedure are given in Section 2. Section 3 demonstrates the applicability of this approach to a real-world vision-based task, rotated handwritten digit recognition. Section 4 closes the paper with conclusions and suggestions for future research. It also briefly describes the second application to which this method has been successfully applied, face detection in cluttered scenes.

## 2  Making Templates Rotationally Invariant

The process to make templates rotationally invariant is easiest to describe in the context of a binary classification problem; the extension to multiple classes is discussed later in this section. Imagine a simplified version of the digit recognition task: we want a detector for a single digit. Suppose we wish to tell whether the input contains the digit '3' or not. The challenge is that the '3' can be rotated within the image plane by an arbitrary amount.

Recognizing rotated objects is a two step process. In the first step, a "De-Rotation" network is applied to the input image. This network analyzes the input before it is given to a "Detection" network. If the input contains a '3', the De-Rotation network returns the digit's angle of rotation. The window can then be rotated by the negative of that angle to make the '3' upright. Note that the De-Rotation network *does not* require a '3' as input. If a non-'3' image is encountered, the De-Rotation network will return an unspecified rotation. However, a rotation of a non-'3' will yield another (perhaps different) image of a non-'3'. When the resulting image is given to the Detection network it will not detect a '3'. On the other hand, a rotated '3', which may not have been detected by the Detection network alone, will be rotated to an upright position by the De-Rotation network, and will subsequently be detected as a '3' by the Detection network.

The Detection network is trained to output a positive value only if the input contains an *upright* '3', and a negative value otherwise (even if it contains a rotated '3'). It should be noted that the methods described here do not require neural networks. As shown in [Le Cun *et al.*, 1995a, Le Cun *et al.*, 1995b] a number of other classifiers can be used.

The De-Rotation and Detection networks are used sequentially. First, the input image is processed by the De-Rotation network which returns an angle of rotation, assuming the image contains a '3'. A simple geometric transformation of the image is performed to undo this rotation. If the original image contained a '3', it would now be upright. The resulting image is then passed to the Detection network. If the original image contained a '3', it can now be successfully detected.

This idea can easily be extended to multiple-class classification problems: a De-Rotation network is trained for each object class to be recognized. For the digit recognition problem, 10 De-Rotation networks are trained, one for each of the digits *0..9*. To classify the digits once they are upright, a single classification network is used with 10 outputs (instead of the detection networks trained on individual digits – alternative approaches will be described later in this paper). The classification network is used in the standard manner; the output with the maximum value is taken as the classification. To classify a new image, the following procedure is used:

**For each digit $D$ $(0 \leq D \leq 9)$:**

1.  Pass image through De-Rotation-network-$D$. This returns the rotation angle.

2.  Rotate the image by (-1.0 * returned rotation angle).

3.  Pass the de-rotated image to the classification network.

4.  If the classification network's maximum output is output $D$, the activation of output $D$ is recorded. Otherwise digit $D$ is eliminated as a candidate.

In most cases, this will eliminate all but one of the candidates. However, in some cases more than one candidate will remain. In these cases, the digit with the maximum recorded activation (from Step 4) is returned. In the unlikely event that no candidates remain, either the system can reject the sample as one it cannot classify, or it can return the maximum value which would have been recorded in Step 4 if none of the examples were rejected.

### 2.1 Network Specifics

To train the De-Rotation networks, images of rotated digits were input, with the rotation angle as the target output. Examples of rotated digits are shown in Figure 2. Each image is 28x28 pixels. The upright data sets are from the MNIST database [Le Cun *et al.*, 1995a].

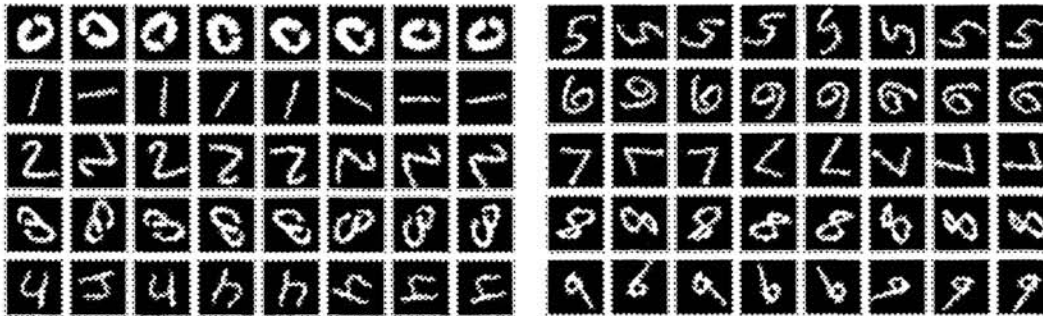

**Figure 2:** 8 examples of each of the 10 digits to be recognized. The first example in each group of eight is shown with no rotation; it is as it appears in the MNIST data set. The second through eighth examples show the same digit rotated in-plane by random amounts.

In the classification network, each output represents a distinct class; therefore, the standard 1-of-$N$ output representation was used with 10 outputs. To represent a continuous variable (the angle of rotation) in the outputs of the De-Rotation network, we used a Gaussian output encoding [Pomerleau, 1992] with 90 output units. With the Gaussian encoding, instead of only training the network to activate a single output (as is done in 1-of-$N$ encoding), outputs close to the desired output are also activated in proportion to their distance from the desired output. This representation avoids the imposed discontinuities of the strict 1-of-$N$ encoding for images which are similar, but have only slight differences in rotations. Further, this representation allows finer granularity with the same number of output units than would be possible if a 1-of-$N$ encoding was used [Pomerleau, 1992].

The network architecture for both the classification and the De-Rotation networks consists of a single hidden layer. However, unlike a standard fully-connected network, each hidden unit was only connected to a small patch of the 28x28 input. The De-Rotation networks used groups of hidden units in which each hidden unit was connected to only 2x2, 3x3, 4x4 & 5x5 patches of the inputs (in each of these groups, the patches were spaced 2x2 pixels apart; therefore, the last three groups had overlapping patches). This is similar to the networks used in [Baluja, 1997][Rowley *et. al*, 1998a, 1998b] for face detection. Unlike the convolution networks used by [Le Cun *et al.*, 1990], the weights into the hidden units were not shared.[1] Note that many different local receptive field configurations were tried; almost all had equivalent performance.

# 3   Rotated Handwritten Digit Recognition

To create a complete rotationally invariant digit recognition system, the first step is to segment each digit from the background. The second is to recognize the digit which has been segmented. Many systems have been proposed for segmenting written digits from background clutter [Jain & Yu, 1997][Sato *et al.*, 1998][Satoh & Kanade, 1997]. In this paper, we concentrate on the recognition portion of the task. Given a segmented image of a potentially rotated digit, how do we recognize the digit?

The first experiment conducted was to establish the base-line performance. We used only the standard, upright training set to train a classification network (this training set consists of 60,000 digits). This network was then tested on the testing set (the testing set contains 10,000 digits). In addition to measuring the performance on the upright testing set, the entire testing set was also rotated. As expected, performance rapidly degrades with rotation. A graph of the performance with respect to the rotation angle is shown in Figure 3.

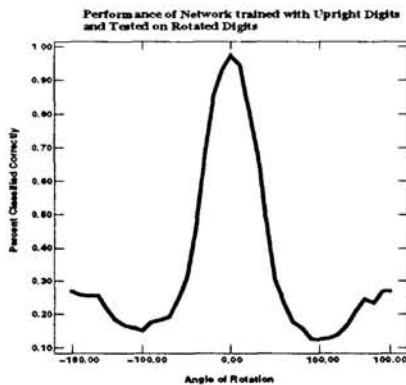

**Figure 3:** Performance of the classification network trained only with upright images when tested on rotated images. As the angle of rotation increases, performance degrades. Note the spike around 180 degrees, this is because some digits look the same even when they are upside-down. The peak performance is approximately 97.5% (when the digits are upright).

It is interesting to note that around 180° rotation, performance slightly rises. This is because some of the digits are symmetric across the center horizontal axis – for example the digits '8', '1', '2' & '5' can be recognized upside-down. Therefore, at these orientations, the upright detector works well for these digits.

As mentioned earlier, the simplest method to make an upright digit classifier handle rotations is to repeatedly rotate the input image and classify it at each rotation. The first drawback to this approach is the severe computational expense. The second drawback is that because the digit is examined at many rotations, it may appear similar to numerous digits in different orientations. One approach to avoid the latter problem is to classify the digit as the one that is voted for most often when examined over all rotations. To ensure that this process is not biased by the size of the increments by which the image is rotated, various angle increments are tried. As shown in the first row of Table I, this method yields low

**Table I:  Exhaustive Search over all possible rotations**

| Exhaustive Search Method | Number of Angle Increments Tried | | |
|---|---|---|---|
| | 360 (1 degree/increment) | 100 (3.6 degree/increment) | 50 (7.2 degrees/increment) |
| Most frequent vote (over all rotations) | 59.5% | 66.0% | 65.0% |
| Most frequent vote – counted only when votes are positive (over all rotations) | 75.2% | 74.5% | 74.0% |

---

1. Note that in the empirical comparisons presented in [Le Cun *et al.*, 1995a], convolution networks performed extremely well in the upright digit recognition task. However, due to limited computation resources, we were unable to train these networks, as each takes 14-20 *days* to train. The network used here was trained in 3 hours, and had approximately a 2.6% misclassification rate on the upright test set. The best networks reported in [Le Cun *et al*, 1995a] have less than 1% error. It should be noted that the De-Rotation networks trained in this study can easily be used in conjunction with any classification procedure, including convolutional networks.

classification accuracies. One reason for this is that a vote is counted even when the classification network predicts all outputs to be less than 0 (the network is trained to predict +1 when a digit is recognized, and -1 when it is not). The above experiment was repeated with the following modification: a vote was only counted when the maximum output of the classification network was above 0. The result is shown in the second row of Table I. The classification rate improved by more than 10%.

Given these base-line performance measures[2], we now have quantitative measurements with which to compare the effectiveness of the approach described in this paper. The performance of the procedure used here, with 10 "De-Rotation" networks and a single classification network, is shown in Figure 4. Note that unlike the graph shown in Figure 3, there is very little effect on the classification performance with the rotation angle.

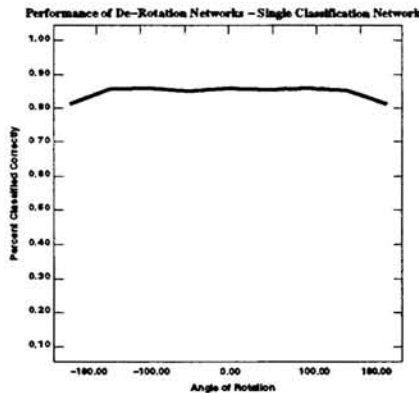

**Figure 4:** Performance of the combined De-Rotation network and classification network system proposed in this paper. Note that the performance is largely unaffected by the rotation. The average performance, over all rotations, is 85.0%.

To provide some intuition of how the De-Rotation networks perform, Figure 5 shows examples of how each De-Rotation networks transform each digit. Each De-Rotation network suggests a rotation which makes the digit look as much like the one with which the network was trained. For example, De-Rotation-Network-5 will suggest a rotation that will make the input digit look as much like the digit '5' as possible; for example, see De-Rotation-Network-5's effect on the digit '4'.

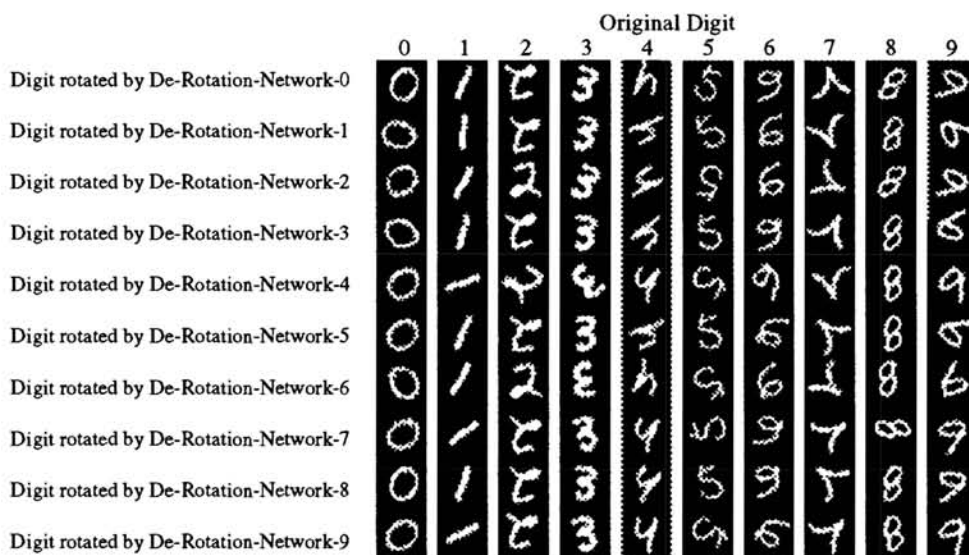

**Figure 5:** Digits which have been rotated by the angles specified by each of the De-rotation networks. As expected (if the method is working), the digits on the diagonal (upper left to bottom right) appear upright.

---

2. Another approach is to train a single network to handle both rotation and classification by using rotated digits as inputs, and the digit's classification as the target output. Experiments with the approach yielded results far below the techniques presented here.

As shown in Figure 4, the average classification accuracy is approximately 85.0%. The performance is not as good as with the upright case alone, which had a peak performance of approximately 97.5% (Figure 3). The high level of performance achieved in the upright case is unlikely for rotated digits: if all rotations are admissible, some characters are ambiguous. The problem is that when working correctly, De-Rotation-Network-*D* will suggest an angle of rotation that will make any input image look as much like the digit *D* as possible through rotation. In most cases when the input image is not the digit *D,* the rotation will not cause the image to look like *D*. However, in some cases, such as those shown in Figure 6(right), the digit will be transformed enough to cause a classification error. Some of these errors will most likely never be correctable (for example, '6' and '9' in some instances); however, there is hope for correcting some of the others.

Figure 6 presents the complete confusion matrix. As can be seen in the examples in Figure 6(right), the digit '4' can be rotated to appear similar to a '5'. Nonetheless, there often remain distinctive features that allow real '5's to be differentiated from the rotated '4's. However, the classification network is unable to make these distinctions because it was not trained with the appropriate examples. Remember, that since the classification network was only trained with the upright digit training set, rotated '4's are never encountered during training. This reflects a fundamental discrepancy in the training/testing procedure. The distributions of images which were used to train the classification network is different than the distributions on which the network is tested.

To address this problem, the classification mechanism is modified. Rather than using the single *1-of-10* neural network classifier used previously, 10 individual Detection networks are used. Each detection network has a single binary output that signifies whether the input contains the digit (upright) with which the network was trained. Each De-Rotation network is paired with the respective Detection network. *The crucial point is that rather than training the Detection-Network-D with the original upright images in the training set, each image (whether it is a positive or negative example) is first passed through De-Rotation-Network-D.* Although this makes training Detection-Network-*D* difficult since all the digits are rotated to appear as much like upright-*D*'s as possible by De-Rotation-Network-*D*, the distribution of training images matches the testing distribution more closely. In use, when a new image is presented, it is passed through the 10 network pairs. Candidate digits are eliminated if the binary output from the detection network does not signal a detection. Preliminary results with this new approach are extremely promising; the classification accuracy increases dramatically – to 93% when averaged over all rotations. This is a more than a 50% reduction in error over the previously described approach.

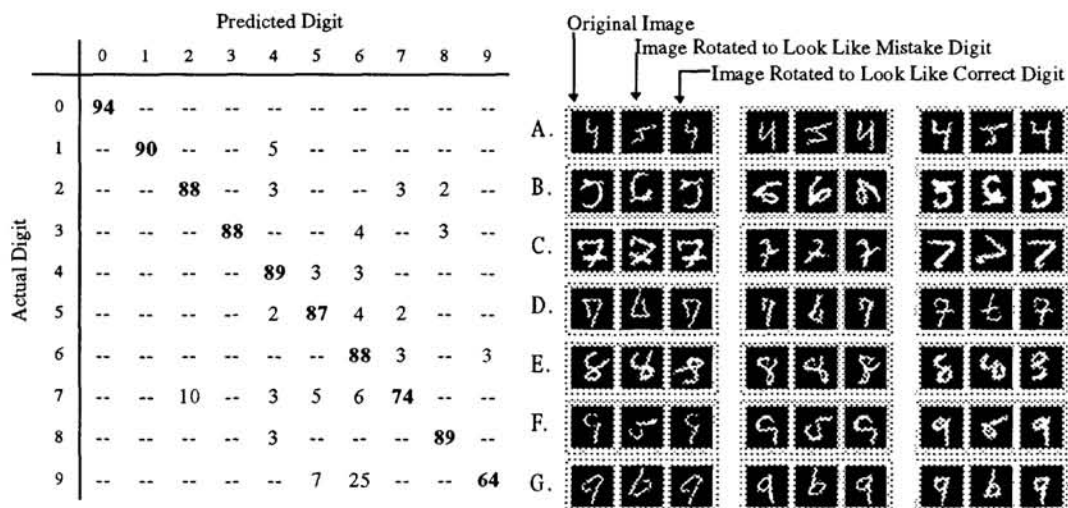

| | | Predicted Digit | | | | | | | | | |
| | | 0 | 1 | 2 | 3 | 4 | 5 | 6 | 7 | 8 | 9 |
|---|---|---|---|---|---|---|---|---|---|---|---|
| Actual Digit | 0 | 94 | -- | -- | -- | -- | -- | -- | -- | -- | -- |
| | 1 | -- | 90 | -- | -- | 5 | -- | -- | -- | -- | -- |
| | 2 | -- | -- | 88 | -- | 3 | -- | -- | 3 | 2 | -- |
| | 3 | -- | -- | -- | 88 | -- | -- | 4 | -- | 3 | -- |
| | 4 | -- | -- | -- | -- | 89 | 3 | 3 | -- | -- | -- |
| | 5 | -- | -- | -- | -- | 2 | 87 | 4 | 2 | -- | -- |
| | 6 | -- | -- | -- | -- | -- | -- | 88 | 3 | -- | 3 |
| | 7 | -- | -- | 10 | -- | 3 | 5 | 6 | 74 | -- | -- |
| | 8 | -- | -- | -- | -- | 3 | -- | -- | -- | 89 | -- |
| | 9 | -- | -- | -- | -- | -- | 7 | 25 | -- | -- | 64 |

**Figure 6:** Example errors. (LEFT) Confusion Matrix (only entries account for 2% or more entries are filled in for ease of reading). (RIGHT) some of the errors made in classification. 3 examples of each of the errors are shown. Row A: '4' mistaken as '5'. Row B: '5' mistaken as '6'. Row C: '7' mistaken as '2'. Row D: '7' mistaken as '6'. Row E: '8' mistaken as '4'. Row F: '9' mistaken as '5'. Row G: '9' mistaken as '6'.

## 4 Conclusions and Future Work

This paper has presented results on the difficult problem of rotated digit recognition. First, we presented base-line results with naive approaches such as exhaustively checking all rotations. These approaches are both slow and have large error rates. Second, we presented results with a novel two-stage approach which is both faster and more effective than the naive approaches. Finally, we presented preliminary results with a new approach that more closely models the training and testing distributions.

We have recently applied the techniques presented in this paper to the detection of faces in cluttered scenes. In previous studies, we presented methods for finding all upright frontal faces [Rowley *et al.*, 1998a]. By using the techniques presented here, we were able to detect all frontal faces, including those which were rotated within the image plane [Baluja, 1997][Rowley *et al.*, 1998b]. The methods presented in this paper should also be directly applicable to full alphabet rotated character recognition.

In this paper, we examined each digit individually. A straight-forward method to eliminate some of the ambiguities between rotationally similar digits is to use contextual information. For example, if surrounding digits are all rotated to the same amount, this provides strong hints about the rotation of nearby digits. Further, in most real-world cases, we might expect digits to be close to upright; therefore, one method of incorporating this information is to penalize matches which rely on large rotation angles.

This paper presented a general way to make template-based recognition rotation invariant. In this study, both the rotation estimation procedures and the recognition templates were implemented with neural-networks. Nonetheless, for classification, any technique which implements a form of templates, such as correlation templates, support vector machines, probabilistic networks, $K$-Nearest Neighbor, or principal component-based methods, could have easily been employed.

## Acknowledgements

The author would like to thank Kaari Flagstad for her reviews of many successive drafts of this paper.

## References

Baluja, S. (1997) "Face Detection with In-Plane Rotation: Early Concepts and Preliminary Results," Justsystem Pittsburgh Research Center Technical Report. JPRC-TR-97-001.

Guyon, I, Poujaud, I., Personnaz, L, Dreyfus, G., Denker, J. LeCun, Y. (1989) "Comparing Different Neural Net Architectures for Classifying Handwritten Digits", in *IJCNN II* 127-132.

Jain, A. & Yu, B. (1997) "Automatic Text Location in Images and Video Frames", TR: MSUCPS: TR 97-33.

Le Cun, Y., Jackel, D., Bottou, L, Cortes, C., Denker, J. Drucker, J. Guyon, I, Miller, U. Sackinger, E. Simard, P. Vapnik, V. (1995a) "Learning Algorithms for Classification: A Comparison on Handwritten Digit Recognition". *Neural Networks: The Statistical Mechanics Perspective*, Oh, J., Kwon, C. & Cho, S. (Ed.), pp. 261-276.

LeCun, Y., Jackel, L. D., Bottou, L., Brunot, A., Cortes, C., Denker, J. S., Drucker, H., Guyon, I., Muller, U. A., Sackinger, E., Simard, P. and Vapnik, V. (1995b), Comparison of learning algorithms for handwritten digit recognition," *ICANN*, Fogelman, F. and Gallinari, P., 1995, pp. 53-60.

LeCun, Y., Boser, B., Denker, J. S., Henderson, D., Howard, R. E., Hubbard, W. and Jackel, L. D. (1990), "Handwritten digit recognition with a back-propagation network," *Advances in Neural Information Processing Systems 2 (NIPS '89)*, Touretzky, David (Ed.), Morgan Kaufman.

Lee, Y. (1991) "Handwritten Digit Recognition using K-NN, RBF and Backpropagation Neural Networks", *Neural Computation*, 3, 3.

Pomerleau, D.A. (1993) *Neural Network Perception for Mobile Robot Guidance*, Kluwer Academic

Rowley, H., Baluja, S. & Kanade, T. (1998a) "Neural Network-Based Face Detection," *IEEE-Transactions on Pattern Analysis and Machine Intelligence (PAMI)*, Vol. 20, No. 1, January, 1998.

Rowley, H., Baluja, S. & Kanade, T. (1998b) "Rotation Invariant Neural Network-Based Face Detection," to appear in *Proceedings of Computer Vision and Pattern Recognition, 1998*.

Sato, T, Kanade, T., Hughes, E. & Smith, M. (1998) "Video OCR for Digital News Archives" to appear in *IEEE International Workshop on Content-Based Access of Image and Video Databases*.

Satoh, S. & Kanade, T. (1997) "Name-It: Association of face and name in Video", in *Proceedings of IEEE Conference on Computer Vision and Pattern Recognition*, 1997.